# New Approaches Towards Robust and Adaptive Speech Recognition

**Hervé Bourlard, Samy Bengio and Katrin Weber**
IDIAP
P.O. Box 592, rue du Simplon 4
1920 Martigny, Switzerland
{*bourlard,bengio,weber*}*@idiap.ch*

## Abstract

In this paper, we discuss some new research directions in automatic speech recognition (ASR), and which somewhat deviate from the usual approaches. More specifically, we will motivate and briefly describe new approaches based on multi-stream and multi/band ASR. These approaches extend the standard hidden Markov model (HMM) based approach by assuming that the different (frequency) channels representing the speech signal are processed by different (independent) "experts", each expert focusing on a different characteristic of the signal, and that the different stream likelihoods (or posteriors) are combined at some (temporal) stage to yield a global recognition output. As a further extension to multi-stream ASR, we will finally introduce a new approach, referred to as HMM2, where the HMM emission probabilities are estimated via state specific feature based HMMs responsible for merging the stream information and modeling their possible correlation.

## 1 Multi-Channel Processing in ASR

Current automatic speech recognition systems are based on (context-dependent or context-independent) phone models described in terms of a sequence of hidden Markov model (HMM) states, where each HMM state is assumed to be characterized by a stationary probability density function. Furthermore, time correlation, and consequently the dynamic of the signal, inside each HMM state is also usually disregarded (although the use of temporal delta and delta-delta features can capture some of this correlation). Consequently, only medium-term dependencies are captured via the topology of the HMM model, while short-term and long-term dependencies are usually very poorly modeled. Ideally, we want to design a particular HMM able to accommodate multiple time-scale characteristics so that we can capture phonetic properties, as well as syllable structures and (long term) invariants that are more robust to noise. It is, however, clear that those different time-scale features will also exhibit different levels of stationarity and will require different HMM topologies to capture their dynamics.

There are many potential advantages to such a multi-stream approach, including:

1. The definition of a principled way to merge different temporal knowledge sources such as acoustic and visual inputs, even if the temporal sequences are not synchronous and do not have the same data rate – see [13] for further discussion about this.
2. Possibility to incorporate multiple time resolutions (as part of a structure with multiple unit lengths, such as phone and syllable).
3. As a particular case of multi-stream processing, multi-band ASR [2, 5], involving the independent processing and combination of partial frequency bands, have many potential advantages briefly discussed below.

In the following, we will not discuss the underlying algorithms (more or less "complex" variants of Viterbi decoding), nor detailed experimental results (see, e.g., [4] for recent results). Instead, we will mainly focus on the combination strategy and discuss different variants arounds the same formalism.

## 2 Multiband-based ASR

### 2.1 General Formalism

As a particular case of the multi-stream paradigm, we have been investigating an ASR approach based on independent processing and combination of frequency subbands. The general idea, as illustrated in Fig. 1, is to split the whole frequency band (represented in terms of critical bands) into a few subbands on which different recognizers are independently applied. The resulting probabilities are then combined for recognition later in the process at some segmental level. Starting from critical bands, acoustic processing is now performed independently for each frequency band, yielding $K$ input streams, each being associated with a particular frequency band.

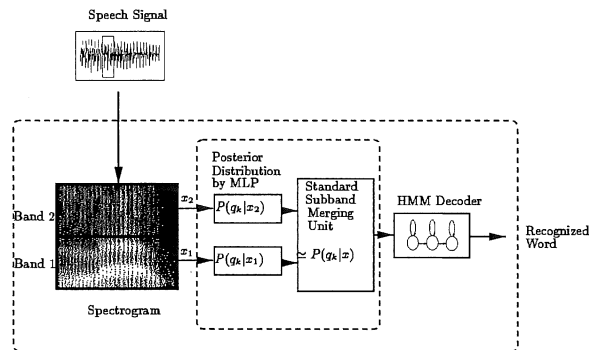

Figure 1: *Typical multiband-based ASR architecture. In multi-band speech recognition, the frequency range is split into several bands, and information in the bands is used for phonetic probability estimation by independent modules. These probabilities are then combined for recognition later in the process at some segmental level.*

In this case, each of the $K$ sub-recognizer (channel) is now using the information contained in a specific frequency band $X^k = \{x_1^k, x_2^k, \ldots, x_n^k, \ldots, x_N^k\}$, where each $x_n^k$ represents the acoustic (spectral) vector at time $n$ in the $k$-th stream.

In the case of hybrid HMM/ANN systems, HMM local emission (posterior) probabilities are estimated by an artificial neural network (ANN), estimating $P(q_j|x_n)$, where $q_j$ is an HMM state and $x_n = (x_n^1, \ldots, x_n^k, \ldots, x_n^K)^t$ the feature vector at time $n$.

In the case of multi-stream (or subband-based) HMM/ANN systems, different ANNs will compute state specific stream posteriors $P(q_j|x_n^k)$. Combination of these local posteriors can then be performed at different temporal levels, and in many ways, including [2]: untrained linear or trained linear (e.g., as a function of automatically estimated local SNR) functions, as well as trained nonlinear functions (e.g., by using a neural network). In the simplest case, this subband posterior recombination is performed at the HMM state level, which then amounts to performing a standard Viterbi decoding in which local (log) probabilities are obtained from a linear or nonlinear combination of the local subband probabilities. For example, in the initial subband-based ASR, local posteriors $P(q_j|x_n)$ were estimated according to:

$$P(q_j|x_n) = \sum_{k=1}^{K} w_k P(q_j|x_n^k, \Theta_k)$$ (1)

where, in our case, each $P(q_j|x_n^k, \Theta_k)$ is computed with a band-specific ANN of parameters $\Theta_k$ and with $x_n^k$ (possibly with temporal context) at its input. The weighting factors can be assigned a uniform distribution (already performing very well [2]) or be proportional to the estimated SNR. Over the last few years, several results were reported showing that such a simple approach was usually more robust to band limited noise.

## 2.2 Motivations and Drawbacks

The multi-band briefly discussed above has several potential advantages summarized here.

**Better robustness to band-limited noise** — The signal may be impaired (e.g., by noise, channel characteristics, reverberation,...) only in some specific frequency bands. When recognition is based on several independent decisions from different frequency subbands, the decoding of a linguistic message need not be severely impaired, as long as the remaining clean subbands supply sufficiently reliable information. This was confirmed by several experiments (see, e.g., [2]). Surprisingly, even when the combination is simply performed at the HMM state level, it is observed that the multi-band approach is yielding better performance and noise robustness than a regular full-band system.

Similar conclusions were also observed in the framework of the missing feature theory [7, 9]. In this case, it was shown that, *if one knows the position of the noisy features*, significantly better classification performance could be achieved by disregarding the noisy data (using marginal distributions) or by integrating over all possible values of the missing data conditionally on the clean features — See Section 3 for further discussion about this.

**Better modeling** — Subband modeling will usually be more robust. Indeed, since the dimension of each (subband) feature space is smaller, it is easier to estimate reliable statistics (resulting in a more robust parametrization). Moreover, the all-pole modeling usually used in ASR will be more robust if performed on subbands, i.e., in lower dimensional spaces, than on the full-band signal [12].

**Channel asynchrony** — Transitions between more stationary segments of speech do not necessarily occur at the same time across the different frequency bands [8], which makes the piecewise stationary assumption more fragile. The subband approach may have the potential of relaxing the synchrony constraint inherent in current HMM systems.

**Channel specific processing and modeling** — Different recognition strate-

gies might ultimately be applied in different subbands. For example, different time/frequency resolution tradeoffs could be chosen (e.g., time resolution and width of analysis window depending on the frequency subband). Finally, some subbands may be inherently better for certain classes of speech sounds than others.

**Major objections and drawbacks** — One of the common objections [8] to this separate modeling of each frequency band has been that important information in the form of correlation between bands may be lost. Although this may be true, several studies [8], as well as the good recognition rates achieved on small frequency bands [3, 6], tend to show that most of the phonetic information is preserved in each frequency band (possibly provided that we have enough temporal information). This drawback will be fixed by the method presented next.

## 3   Full Combination Subband ASR

If we know where the noise is, and based on the results obtained with missing data [7, 9], impressive noise robustness can be achieved by using the marginal distribution, estimating the HMM emission probability based on the clean frequency bands only. In our subband approach, we do not assume that we know, or detect explicitly, where the noise is. Following the above developments and discussions, it thus seems reasonable to integrate over all possible positions of the noisy bands, and thus to simultaneously deal with all the $L = 2^K$ possible subband combinations $S_n^\ell$ (with $\ell = 1, \ldots, L$, and also including the empty set) extracted from the feature vector $x_n$. Introducing the hidden variable $E_n^\ell$, representing the statistical (exclusive and mutually exhaustive) event that the feature subset $S_n^\ell$ is "clean" (reliable), and integrating over all its possible values, we can then rewrite the local posterior probability as:

$$
\begin{aligned}
P(q_j|x_n, \Theta) = \sum_{\ell=1}^{L} P(q_j, E_n^\ell|x_n, \Theta) &= \sum_{\ell=1}^{L} P(q_j|E_n^\ell, x_n, \Theta_\ell) P(E_n^\ell|x_n) \\
&= \sum_{\ell=1}^{L} P(q_j|S_n^\ell, \Theta_\ell) P(E_n^\ell|x_n) \qquad (2)
\end{aligned}
$$

where $P(E_n^\ell|x_n)$ represents the relative reliability of a specific feature set. $\Theta$ represents the whole parameter space, while $\Theta_\ell$ denotes the set of (ANN) parameters used to compute the subband posteriors.

Typically, training of the $L$ neural nets would be done once and for all on clean data, and the recognizer would then be *adapted* on line simply by adjusting the weights $P(E_n^\ell|x_n)$ (still representing a limited set of $L$ parameters) to increase the global posteriors. This adaptation can be performed by online estimation of the signal-to-noise ratio or by online, unsupervised, EM adaptation.

While it is pretty easy to quickly estimate any subband likelihood or marginal distribution when working with Gaussian or multi-Gaussian densities [7], straigh implementation of (2) is not always tractable since it requires the use (and training) of $L$ neural networks to estimate all the posteriors $P(q_j|S_n^\ell, \Theta_\ell)$. However, it has the advantage of not requiring the subband independence assumption [3].

An interesting approximation to this "optimal" solution though consists in simply using the neural nets that are available ($K$ of them in the case of baseline subband ASR) and, re-introducing the independence assumption, to approximate all the

other subband combination probabilities in (2), as follows [3, 4]:

$$P(q_j|S_n^\ell, \Theta_\ell) = P(q_j) \prod_{k \in S^\ell} \frac{P(q_j|x_n^k, \Theta_k)}{P(q_j)} \tag{3}$$

Experimental results obtained from this Full Combination approach in different noisy conditions are reported in [3, 4], where the performance of this above approximation was also compared to the "optimal" estimators (2). Interestingly, it was shown that this independence assumption did not hurt much and that the resulting recognition performance was similar to the performance obtained by training and recombining all possible $L$ nets (and significantly better than the original subband approach). In both cases, the recognition rate and the robustness to noise were greatly improved compared to the initial subband approach. This further confirms that we do not seem to lose "critically" important information when neglecting the correlation between bands.

In the next section, we biefly introduced a further extension of this approach where the segmentation into subbands is no longer done explicitly, but is achieved dynamically over time, and where the integration over all possible frequency segmentation is part of the same formalism.

## 4   HMM2: Mixture of HMMs

HMM emission probabilities are typically modeled through Gaussian mixtures or neural networks. We propose here an alternative approach, referred to as HMM2, integrating standard HMMs (referred to as "temporal HMMs") with state-dependent feature-based HMMs (referred to as "feature HMMs") responsible for the estimation of the emission probabilities. In this case, each feature vector $x_n$ at time $n$ is considered as a fixed length sequence, which has supposedly been generated by a temporal HMM state specific HMM for which each state is emitting individual feature components that are modeled by, e.g., one dimensional Gaussian mixtures. The feature HMM thus looks at all possible subband segmentations and automatically performs the combination of the likelihoods to yield a single emission probability.

The resulting architecture is illustrated in Figure 2. In this example, the HMM2 is composed of an HMM that handle sequences of features through time. This HMM is composed of 3 *left-to-right* connected states ($q_1$, $q_2$ and $q_3$) and each state emits a vector of features at each time step. The particularity of an HMM2 is that each state uses an HMM to emit the feature vector, as if it was an ordered sequence (instead of a vector). In Figure 2, state $q_2$ contains a feature HMM with 4 states connected *top-down*. Of course, while the temporal HMM usually has a left-to-right structure, the topology of the feature HMM can take many forms, which will then reflect the correlation being captured by the model. The feature HMM could even have more states than feature components, in which case "high-order" correlation information could be extracted.

In [1], an EM algorithm to jointly train all the parameters of such HMM2 in order to maximize the data likelihood has been derived. This derivation was based on the fact that an HMM2 can be considered as a mixture of mixtures of distributions.

We believe that HMM2 (which includes the classical mixture of Gaussian HMMs as a particular case) has several potential advantages, including:

1. Better feature correlation modeling through the feature-based (frequency) HMM topology. Also, the complexity of this topology and the probability

density function associated with each state easily control the number of parameters.

2. Automatic non-linear spectral warping. In the same way the conventional HMM does time warping and time integration, the feature-based HMM performs frequency warping and frequency integration.

3. Dynamic formant trajectory modelling. As further discussed below, the HMM2 structure has the potential to extract some relevant formant structure information, which is often considered as important to robust speech recognition.

To illustrate the last point and its relationship with dynamic multi-band ASR, the HMM2 models was used in [14] to extract formant-like information. All the parameters of HMM2 models were trained according to the above EM algorithm on delta-frequency features (differences of two consecutive log Rasta PLP coefficients). The feature HMM had a simple top-down topology with 4 states. After training, Figure 3 shows (on unseen test data) the value of the features for the phoneme *iy* as well as the segmentation found by a Viterbi decoding along the delta-frequency axis (the thick black lines). At each time step, we kept the 3 positions where the delta-frequency HMM changed its state during decoding (for instance, at the first time frame, the HMM goes from state 1 to state 2 after the third feature). We believe they contain formant-like information. In [14], it has been shown that the use of that information could significantly enhance standard speech recognition systems.

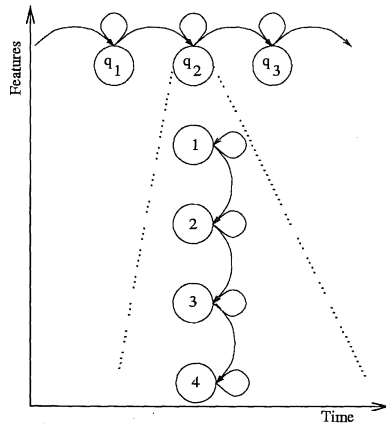

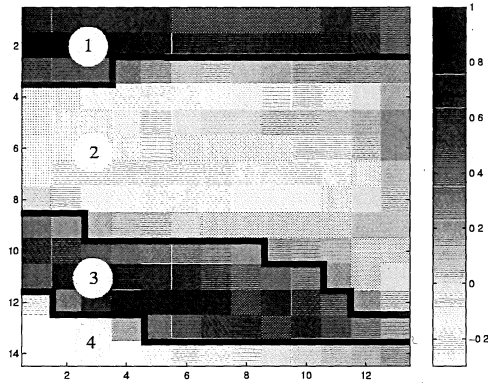

Figure 2: An HMM2: the emission distributions of the HMM are estimated by another HMM.

Figure 3: Frequency deltas of log Rasta PLP and segmentation for an example of phoneme *iy*.

**Acknowledgments**

The content and themes discussed in this paper largely benefited from the collaboration with our colleagues Andrew Morris, Astrid Hagen and Hervé Glotin. This work was partly supported by the Swiss Federal Office for Education and Science (FOES) through the European SPHEAR (TMR, Training and Mobility of Researchers) and RESPITE (ESPRIT Long term Research) projects. Additionnally, Katrin Weber is supported by a the Swiss National Science Foundation project MULTICHAN.

# References

[1] Bengio, S., Bourlard, H., and Weber, K., "An EM Algorithm for HMMs with Emission Distributions Represented by HMMs," *IDIAP Research Report*, IDIAP-RR00-11, Martigny, Switzerland, 2000.

[2] Bourlard, H. and Dupont, S., "A new ASR approach based on independent processing and combination of partial frequency bands," *Proc. of Intl. Conf. on Spoken Language Processing* (Philadelphia), pp. 422-425, October 1996.

[3] Hagen, A., Morris, A., Bourlard, H., "Subband-based speech recognition in noisy conditions: The full combination approach," *IDIAP Research Report no. IDIAP-RR-98-15*, 1998.

[4] Hagen, A., Morris, A., Bourlard, H., "Different weighting schemes in the full combination subbands approach for noise robust ASR," *Proceedings of the Workshop on Robust Methods for Speech Recognition in Adverse Conditions* (Tampere, Finland), May 25-26, 1999.

[5] Hermansky, H., Pavel, M., and Tribewala, S., "Towards ASR using partially corrupted speech," *Proc. of Intl. Conf. on Spoken Language Processing* (Philadelphia), pp. 458-461, October 1996.

[6] Hermansky, H. and Sharma, S., "Temporal patterns (TRAPS) in ASR noisy speech," *Proc. of the IEEE Intl. Conf. on Acoustics, Speech, and Signal Processing* (Phoenix, AZ), pp. 289-292, March 1999.

[7] Lippmann, R.P., Carlson, B.A., "Using missing feature theory to actively select features for robust speech recognition with interruptions, filtering and noise," *Proc. Eurospeech'97* (Rhodes, Greece, September 1997), pp. KN37-40.

[8] Mirghafori, N. and Morgan, N., "Transmissions and transitions: A study of two common assumptions in multi-band ASR," *Intl. IEEE Conf. on Acoustics, Speech, and Signal Processing*, (Seattle, WA, May 1997), pp. 713-716.

[9] Morris, A.C., Cooke, M.P., and Green, P.D., "Some solutions to the missing features problem in data classification, with application to noise robust ASR," *Proc. Intl. Conf on Acoustics, Speech, and Signal Processing*, pp. 737-740, 1998.

[10] Morris, A.C., Hagen, A., Bourlard, H., "The full combination subbands approach to noise robust HMM/ANN-based ASR," *Proc. of Eurospeech'99* (Budapest, Sep. 99).

[11] Okawa, S., Bocchieri, E., Potamianos, A., "Multi-band speech recognition in noisy environment," *Proc. IEEE Intl. Conf. on Acoustics, Speech, and Signal Processing*, 1998.

[12] Rao, S. and Pearlman, W.A., "Analysis of linear prediction, coding, and spectral estimation from subbands," *IEEE Trans. on Information Theory*, vol. 42, pp. 1160–1178, July 1996.

[13] Tomlinson, M.J., Russel, M.J., Moore, R.K., Bucklan, A.P., and Fawley, M.A., "Modelling asynchrony in speech using elementary single-signal decomposition," *Proc. of IEEE Intl. Conf. on Acoustics, Speech, and Signal Processing* (Munich), pp. 1247-1250, April 1997.

[14] Weber, K., Bengio, S., and Bourlard, H., "HMM2–Extraction of Formant Features and their Use for Robust ASR," *IDIAP Research Report*, IDIAP-RR00-42, Martigny, Switzerland, 2000.

